# Truly Nonparametric Online Variational Inference for Hierarchical Dirichlet Processes

**Michael Bryant**  and  **Erik B. Sudderth**
Department of Computer Science, Brown University, Providence, RI
mbryantj@gmail.com, sudderth@cs.brown.edu

## Abstract

Variational methods provide a computationally scalable alternative to Monte Carlo methods for large-scale, Bayesian nonparametric learning. In practice, however, conventional batch and online variational methods quickly become trapped in local optima. In this paper, we consider a nonparametric topic model based on the hierarchical Dirichlet process (HDP), and develop a novel online variational inference algorithm based on split-merge topic updates. We derive a simpler and faster variational approximation of the HDP, and show that by intelligently splitting and merging components of the variational posterior, we can achieve substantially better predictions of test data than conventional online and batch variational algorithms. For streaming analysis of large datasets where batch analysis is infeasible, we show that our split-merge updates better capture the nonparametric properties of the underlying model, allowing continual learning of new topics.

## 1 Introduction

Bayesian nonparametric methods provide an increasingly important framework for unsupervised learning from structured data. For example, the *hierarchical Dirichlet process* (HDP) [1] provides a general approach to joint clustering of grouped data, and leads to effective nonparametric topic models. While nonparametric methods are best motivated by their potential to capture the details of large datasets, practical applications have been limited by the poor computational scaling of conventional Monte Carlo learning algorithms.

Mean field variational methods provide an alternative, optimization-based framework for nonparametric learning [2, 3]. Aiming at larger-scale applications, recent work [4] has extended online variational methods [5] for the parametric, *latent Dirichlet allocation* (LDA) topic model [6] to the HDP. While this online approach can produce reasonable models of large data streams, we show that the variational posteriors of existing algorithms often converge to poor local optima. Multiple runs are usually necessary to show robust performance, reducing the desired computational gains. Furthermore, by applying a fixed truncation to the number of posterior topics or clusters, conventional variational methods limit the ability of purportedly nonparametric models to fully adapt to the data.

In this paper, we propose novel split-merge moves for online variational inference for the HDP (oHDP) which result in much better predictive performance. We validate our approach on two corpora, one with millions of documents. We also propose an alternative, direct assignment HDP representation which is faster and more accurate than the Chinese restaurant franchise representation used in prior work [4]. Additionally, the inclusion of split-merge moves during posterior inference allows us to dynamically vary the truncation level throughout learning. While conservative truncations can be theoretically justifed for batch analysis of fixed-size datasets [2], our data-driven adaptation of the trunction level is far better suited to large-scale analysis of streaming data.

Split-merge proposals have been previously investigated for Monte Carlo analysis of nonparametric models [7, 8, 9]. They have also been used for maximum likelihood and variational analysis of

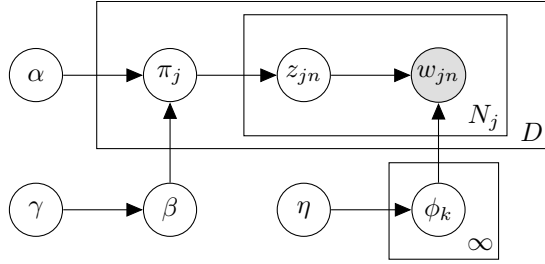

Figure 1: Directed graphical representation of a hierarchical Dirichlet process topic model, in which an unbounded collection of topics $\phi_k$ model the $N_j$ words in each of $D$ documents. Topics occur with frequency $\pi_j$ in document $j$, and with frequency $\beta$ across the full corpus.

parametric models [10, 11, 12, 13]. These deterministic algorithms validate split-merge proposals by evaluating a batch objective on the entire dataset, an approach which is unexplored for nonparametric models and infeasible for online learning. We instead optimize the variational objective via stochastic gradient ascent, and split or merge based on only a noisy estimate of the variational lower bound. Over time, these local decisions lead to global estimates of the number of topics present in a given corpus. We review the HDP and conventional variational methods in Sec. 2, develop our novel split-merge procedure in Sec. 3, and evaluate on various document corpora in Sec. 4.

## 2 Variational Inference for Bayesian Nonparametric Models

### 2.1 Hierarchical Dirichlet processes

The HDP is a hierarchical nonparametric prior for grouped mixed-membership data. In its simplest form, it consists of a top-level DP and a collection of $D$ bottom-level DPs (indexed by $j$) which share the top-level DP as their base measure:

$$G_0 \sim \mathrm{DP}(\gamma H), \qquad G_j \sim \mathrm{DP}(\alpha G_0), \quad j = 1, \ldots, D.$$

Here, $H$ is a base measure on some parameter space, and $\gamma > 0$, $\alpha > 0$ are concentration parameters. Using a stick-breaking representation [1] of the global measure $G_0$, the HDP can be expressed as

$$G_0 = \sum_{k=1}^{\infty} \beta_k \delta_{\phi_k}, \qquad G_j = \sum_{k=1}^{\infty} \pi_{jk} \delta_{\phi_k}.$$

The global weights $\beta$ are drawn from a stick-breaking distribution $\beta \sim \mathrm{GEM}(\gamma)$, and atoms are independently drawn as $\phi_k \sim H$. Each $G_j$ shares atoms with the global measure $G$, and the lower-level weights are drawn $\pi_j \sim \mathrm{DP}(\alpha\beta)$. For this *direct assignment* representation, the $k$ indices for each $G_j$ index directly into the global set of atoms. To complete the definition of the general HDP, parameters $\psi_{jn} \sim G_j$ are then drawn for each observation $n$ in group $j$, and observations are drawn $x_{jn} \sim F(\psi_{jn})$ for some likelihood family $F$. Note that $\psi_{jn} = \phi_{z_{jn}}$ for some discrete indicator $z_{jn}$.

In this paper we focus on an application of the HDP to modeling document corpora. The *topics* $\phi_k \sim \mathrm{Dirichlet}(\eta)$ are distributions on a vocabulary of $W$ words. The global topic weights, $\beta \sim \mathrm{GEM}(\gamma)$, are still drawn from a stick-breaking prior. For each document $j$, document-specific topic frequencies are drawn $\pi_j \sim \mathrm{DP}(\alpha\beta)$. Then for each word index $n$ in document $j$, a topic indicator is drawn $z_{jn} \sim \mathrm{Categorical}(\pi_j)$, and finally a word is drawn $w_{jn} \sim \mathrm{Categorical}(\phi_{z_{jn}})$.

### 2.2 Batch Variational Inference for the HDP

We use variational inference [14] to approximate the posterior of the latent variables $(\boldsymbol{\phi}, \beta, \boldsymbol{\pi}, \mathbf{z})$ — the topics, global topic weights, document-specific topic weights, and topic indicators, respectively — with a tractable distribution $q$, indexed by a set of free variational parameters. Appealing to mean field methods, our variational distribution is fully factorized, and is of the form

$$q(\boldsymbol{\phi}, \beta, \boldsymbol{\pi}, \mathbf{z} \mid \boldsymbol{\lambda}, \boldsymbol{\theta}, \boldsymbol{\varphi}) = q(\beta) \prod_{k=1}^{\infty} q(\phi_k \mid \lambda_k) \prod_{j=1}^{D} q(\pi_j \mid \theta_j) \prod_{n=1}^{N_j} q(z_{jn} \mid \varphi_{jn}), \qquad (1)$$

where $D$ is the number of documents in the corpus and $N_j$ is the number of words in document $j$. Individual distributions are selected from appropriate exponential families:

$$q(\beta) = \delta_{\beta^*}(\beta)$$
$$q(\phi_k \mid \lambda_k) = \text{Dirichlet}(\phi_k \mid \lambda_k)$$
$$q(\pi_j \mid \theta_j) = \text{Dirichlet}(\pi_j \mid \theta_j)$$
$$q(z_{jn}) = \text{Categorical}(z_{jn} \mid \varphi_{jn})$$

where $\delta_{\beta^*}(\beta)$ denotes a degenerate distribution at the point $\beta^*$.[1] In our update derivations below, we use $\varphi_{jw}$ to denote the shared $\varphi_{jn}$ for all word tokens in document $j$ of type $w$.

Selection of an appropriate truncation strategy is crucial to the accuracy of variational methods for nonparametric models. Here, we truncate the topic indicator distributions by fixing $q(z_{jn} = k) = 0$ for $k > K$, where $K$ is a threshold which varies *dynamically* in our later algorithms. With this assumption, the topic distributions with indices greater than $K$ are conditionally independent of the observed data; we may thus ignore them and tractably update the remaining parameters with respect to the true, infinite model. A similar truncation has been previously used in the context of an otherwise more complex collapsed variational method [3]. Desirably, this truncation is nested such that increasing $K$ always gives potentially improved bounds, but does not require the computation of infinite sums, as in [16]. In contrast, approximations based on truncations of the stick-breaking topic frequency prior [2, 4] are not nested, and their artifactual placement of extra mass on the final topic $K$ is less suitable for our split-merge online variational inference.

Via standard convexity arguments [14], we lower bound the marginal log likelihood of the observed data using the expected complete-data log likelihood and the entropy of the variational distribution,

$$\mathcal{L}(q) \overset{\text{def}}{=} \mathbb{E}_q[\log p(\boldsymbol{\phi}, \beta, \boldsymbol{\pi}, \mathbf{z}, \mathbf{w} \mid \alpha, \gamma, \eta)] - \mathbb{E}_q[\log q(\boldsymbol{\phi}, \boldsymbol{\pi}, \mathbf{z} \mid \boldsymbol{\lambda}, \boldsymbol{\theta}, \boldsymbol{\varphi})]$$
$$= \mathbb{E}_q[\log p(\mathbf{w} \mid \mathbf{z}, \boldsymbol{\phi})] + \mathbb{E}_q[\log p(\mathbf{z} \mid \boldsymbol{\pi})] + \mathbb{E}_q[\log p(\boldsymbol{\pi} \mid \alpha\beta)] + \mathbb{E}_q[\log p(\boldsymbol{\phi} \mid \eta)]$$
$$+ \mathbb{E}_q[\log p(\beta \mid \gamma)] - \mathbb{E}_q[\log q(\mathbf{z} \mid \boldsymbol{\varphi})] - \mathbb{E}_q[\log q(\boldsymbol{\pi} \mid \boldsymbol{\theta})] - \mathbb{E}_q[\log q(\boldsymbol{\phi} \mid \boldsymbol{\lambda})]$$
$$= \sum_{j=1}^{D} \Big\{ \mathbb{E}_q[\log p(\mathbf{w}_j \mid \mathbf{z}_j, \boldsymbol{\phi})] + \mathbb{E}_q[\log p(\mathbf{z}_j \mid \pi_j)] + \mathbb{E}_q[\log p(\pi_j \mid \alpha\beta)] - \mathbb{E}_q[\log q(\mathbf{z}_j \mid \boldsymbol{\varphi}_j)]$$
$$- \mathbb{E}_q[\log q(\pi_j \mid \theta_j)] + \frac{1}{D}\Big( \mathbb{E}_q[\log p(\boldsymbol{\phi} \mid \eta)] + \mathbb{E}_q[\log p(\beta \mid \gamma)] - \mathbb{E}_q[\log q(\boldsymbol{\phi} \mid \boldsymbol{\lambda})]\Big)\Big\}, \quad (2)$$

and maximize this quantity by coordinate ascent on the variational parameters. The expectations are with respect to the variational distribution. Each expectation is dependent on only a subset of the variational parameters; we leave off particular subscripts for notational clarity. Note that the expansion of the variational lower bound in (2) contains all terms inside a summation over documents. This is the key observation that allowed [5] to develop an online inference algorithm for LDA. A full expansion of the variational objective is given in the supplemental material. Taking derivatives of $\mathcal{L}(q)$ with respect to each of the variational parameters yields the following updates:

$$\varphi_{jwk} \propto \exp\left\{\mathbb{E}_q[\log \phi_{kw}] + \mathbb{E}_q[\log \pi_{jk}]\right\} \quad (3)$$
$$\theta_{jk} \leftarrow \alpha\beta_k + \sum_{w=1}^{W} n_{w(j)}\varphi_{jwk} \quad (4)$$
$$\lambda_{kw} \leftarrow \eta + \sum_{j=1}^{D} n_{w(j)}\varphi_{jwk}, \quad (5)$$

Here, $n_{w(j)}$ is the number of times word $w$ appears in document $j$. The expectations in (3) are

$$\mathbb{E}_q[\log \phi_{kw}] = \Psi(\lambda_{kw}) - \Psi(\textstyle\sum_i \lambda_{ki}), \qquad \mathbb{E}_q[\log \pi_{jk}] = \Psi(\theta_{jk}) - \Psi(\textstyle\sum_i \theta_{ji}),$$

where $\Psi(x)$ is the digamma function, the first derivative of the log of the gamma function.

In evaluating our objective, we represent $\beta^*$ as a $(K+1)$-dim. vector containing the probabilities of the first $K$ topics, and the total mass of all other topics. While $\beta^*$ cannot be optimized in closed form, it can be updated via gradient-based methods; we use a variant of L-BFGS. Drawing a parallel between variational inference and the expectation maximization (EM) algorithm, we label the document-specific updates of $(\boldsymbol{\varphi}_j, \theta_j)$ the E-step, and the corpus-wide updates of $(\boldsymbol{\lambda}, \beta)$ the M-step.

## 2.3 Online Variational Inference

Batch variational inference requires a full pass through the data at each iteration, making it computationally infeasible for large datasets and impossible for streaming data. To remedy this, we adapt and improve recent work on online variational inference algorithms [4, 5].

The form of the lower bound in (2), as a scaled expectation with respect to the document collection, suggests an online learning algorithm. Given a learning rate $\rho_t$ satisfying $\sum_{t=0}^{\infty} \rho_t = \infty$ and $\sum_{t=0}^{\infty} \rho_t^2 < \infty$, we can optimize the variational objective stochastically. Each update begins by sampling a "mini-batch" of documents $\mathcal{S}$, of size $|\mathcal{S}|$. After updating the mini-batch of document-specific parameters $(\varphi_j, \theta_j)$ by iterating (3,4), we update the corpus-wide parameters as

$$\lambda_{kw} \leftarrow (1 - \rho_t)\lambda_{kw} + \rho_t \hat{\lambda}_{kw}, \tag{6}$$

$$\beta_k^* \leftarrow (1 - \rho_t)\beta_k^* + \rho_t \hat{\beta}_k, \tag{7}$$

where $\hat{\lambda}_{kw}$ is a set of *sufficient statistics* for topic $k$, computed from a noisy estimate of (5):

$$\hat{\lambda}_{kw} = \eta + \frac{D}{|\mathcal{S}|} \sum_{j \in \mathcal{S}} n_{w(j)} \varphi_{jwk}. \tag{8}$$

The candidate topic weights $\hat{\beta}$ are found via gradient-based optimization on $\mathcal{S}$. The resulting inference algorithm is similar to conventional batch methods, but is applicable to streaming, big data.

## 3 Split-Merge Updates for Online Variational Inference

We develop a data-driven split-merge algorithm for online variational inference for the HDP, referred to as oHDP-SM. The algorithm dynamically expands and contracts the truncation level $K$ by splitting and merging topics during specialized moves which are interleaved with standard online variational updates. The resulting model truly allows the number of topics to grow with the data. As such, we do not have to employ the technique of [4, 3] and other truncated variational approaches of setting $K$ above the expected number of topics and relying on the inference to infer a smaller number. Instead, we initialize with small $K$ and let the inference discover new topics as it progresses, similar to the approach used in [17]. One can see how this property would be desirable in an online setting, as documents seen after many inference steps may still create new topics.

### 3.1 Split: Creation of New Topics

Given the result of analyzing one mini-batch $q^* = \left\{ (\varphi_j, \theta_j)_{j=1}^{|\mathcal{S}|}, \boldsymbol{\lambda}, \beta^* \right\}$, and the corresponding value of the lower bound $\mathcal{L}(q^*)$, we consider splitting topic $k$ into two topics $k'$, $k''$.[2] The split procedure proceeds as follows: (1) initialize all variational posteriors to break symmetry between the new topics, using information from the data; (2) refine the new variational posteriors using a *restricted iteration*; (3) accept or reject the split via the change in variational objective value.

**Initialize new variational posteriors** To break symmetry, we initialize the new topic posteriors $(\lambda_{k'}, \lambda_{k''})$, and topic weights $(\beta_{k'}^*, \beta_{k''}^*)$, using sufficient statistics from the previous iteration:

$$\lambda_{k'} = (1 - \rho_t)\lambda_k, \qquad\qquad \lambda_{k''} = \rho_t \hat{\lambda}_k,$$
$$\beta_{k'}^* = (1 - \rho_t)\beta_k^*, \qquad\qquad \beta_{k''}^* = \rho_t \hat{\beta}_k.$$

Intuitively, we expect the sufficient statistics to provide insight into how a topic was actually *used* during the E-step. The minibatch-specific parameters $\{\varphi_j, \theta_j\}_{j=1}^{|\mathcal{S}|}$ are then initialized as follows,

$$\varphi_{jwk'} = \omega_k \varphi_{jwk}, \qquad\qquad \varphi_{jwk''} = (1 - \omega_k)\varphi_{jwk},$$
$$\theta_{jk'} = \omega_k \theta_{jk}, \qquad\qquad \theta_{jk''} = (1 - \omega_k)\theta_{jk},$$

with the weights defined as $\omega_k = \beta_{k'}/(\beta_{k'} + \beta_{k''})$.

**Algorithm 1** Restricted iteration
---
1: initialize $(\lambda_\ell, \beta_\ell)$ for $\ell \in \{k', k''\}$
2: **for** $j \in \mathcal{S}$ **do**
3:    initialize $(\varphi_j, \theta_j)$ for $\ell \in \{k', k''\}$
4:    **while** not converged **do**
5:       update $(\varphi_j, \theta_j)$ for $\ell \in \{k', k''\}$ using (3, 4)
6:    **end while**
7:    update $(\lambda_\ell, \beta_\ell)$ for $\ell \in \{k', k''\}$ using (6, 7)
8: **end for**
---

**Restricted iteration**    After initializing the variational parameters for the new topics, we update them through a restricted iteration of online variational inference. The restricted iteration consists of restricted analogues to both the E-step and the M-step, where all parameters except those for the new topics are held constant. This procedure is similar to, and inspired by, the "partial E-step" for split-merge EM [10] and restricted Gibbs updates for split-merge MCMC methods [7].

All values of $\varphi_{jw\ell}$ and $\theta_{j\ell}, \ell \notin \{k', k''\}$, remain unchanged. It is important to note that even though these values are not updated, they are still used in the calculations for both the variational expectation of $\pi_j$ and the normalization of $\varphi$. In particular,

$$\varphi_{jwk'} = \frac{\exp\left\{\mathbb{E}_q[\log\phi_{k'w}] + \mathbb{E}_q[\log\pi_{jk'}]\right\}}{\sum_{\ell\in\mathcal{T}} \exp\left\{\mathbb{E}_q[\log\phi_{\ell w}] + \mathbb{E}_q[\log\pi_{j\ell}]\right\}},$$
$$\mathbb{E}_q[\log\pi_{jk'}] = \Psi(\theta_{jk'}) - \Psi(\textstyle\sum_{k\in\mathcal{T}}\theta_{jk}),$$

where $\mathcal{T}$ is the original set of topics, minus $k$, plus $k'$ and $k''$. The expected log word probabilities $\mathbb{E}_q[\log\phi_{k'w}]$ and $\mathbb{E}_q[\log\phi_{k''w}]$ are computed using the newly updated $\lambda$ values.

**Evaluate Split Quality**    Let $\varphi_{\text{split}}$ for minibatch $\mathcal{S}$ be $\varphi$ as defined above, but with $\varphi_{jwk}$ replaced by the $\varphi_{jwk'}$ and $\varphi_{jwk''}$ learned in the restricted E-step. Let $\theta_{\text{split}}, \lambda_{\text{split}}$ and $\beta^*_{\text{split}}$ be defined similarly. Now we have a new model state $q^{\text{split}(k)} = \left\{(\varphi_{\text{split}}, \theta_{\text{split}})^{|\mathcal{S}|}_{j=1}, \lambda_{\text{split}}, \beta^*_{\text{split}}\right\}$. We calculate $\mathcal{L}\left(q^{\text{split}(k)}\right)$, and if $\mathcal{L}\left(q^{\text{split}(k)}\right) > \mathcal{L}(q^*)$, we update the new model state $q^* \leftarrow q^{\text{split}(k)}$, accepting the split. If $\mathcal{L}\left(q^{\text{split}(k)}\right) < \mathcal{L}(q^*)$, then we go back and test another split, until all splits are tested. In practice we limit the maximum number of allowed splits each iteration to a small constant. If we wish to allow the model to expand the number of topics more quickly, we can increase this number. Finally, it is important to note that all aspects of the split procedure are driven by the data — the new topics are initialized using data-driven proposals, refined by re-running the variational E-step, and accepted based on an unbiased estimate of the change in the variational objective.

### 3.2 Merge: Removal of Redundant Topics

Consider a candidate merge of two topics, $k'$ and $k''$, into a new topic $k$. For batch variational methods, it is straightforward to determine whether such a merge will increase or decrease the variational objective by combining all parameters for all documents,

$$\varphi_{jwk} = \varphi_{jwk'} + \varphi_{jwk''}, \qquad \theta_{jk} = \theta_{jk'} + \theta_{jk''}, \qquad \beta_k = \beta_{k'} + \beta_{k''}, \qquad \lambda_k = \lambda_{k'} + \lambda_{k''},$$

and computing the difference in the variational objective before and after the merge. Because many terms cancel, computing this bound change is fairly computationally inexpensive, but it can still be computationally infeasible to consider all pairs of topics for large $K$. Instead, we identify potential merge candidates by looking at the sample covariance of the $\theta_j$ vectors across the corpus (or minibatch). Topics with positive covariance above a certain threshold have the quantitative effects of their merge evaluated. Intuitively, if there are two copies of a topic or a topic is split into two pieces, they should tend to be used together, and therefore have positive covariance. For consistency in notation, we call the model state with topics $k'$ and $k''$ merged $q^{\text{merge}(k',k'')}$.

Combining this merge procedure with the previous split proposals leads to the online variational method of Algorithm 2. In an online setting, we can only compute unbiased noisy estimates of the true difference in the variational objective; split or merge moves that increase the *expected* variational objective are not guaranteed to do so for the objective evaluated over the entire corpus. The

---
**Algorithm 2** Online variational inference for the HDP + split-merge
---
1: initialize $(\boldsymbol{\lambda}, \beta^*)$
2: **for** $t = 1, 2, \ldots$ **do**
3:     **for** $j \in$ minibatch $\mathcal{S}$ **do**
4:         initialize $(\boldsymbol{\varphi}_j, \theta_j)$
5:         **while** not converged **do**
6:             update $(\boldsymbol{\varphi}_j, \theta_j)$ using (3, 4)
7:         **end while**
8:     **end for**
9:     **for** pairs of topics $\{k', k''\} \in K \times K$ with $\mathrm{Cov}(\theta_{jk'}, \theta_{jk''}) > 0$ **do**
10:         **if** $\mathcal{L}\big(q^{\mathrm{merge}(k', k'')}\big) > \mathcal{L}(q)$ **then**
11:             $q \leftarrow q^{\mathrm{merge}(k', k'')}$
12:         **end if**
13:     **end for**
14:     update $(\boldsymbol{\lambda}, \beta^*)$ using (6, 7)
15:     **for** $k = 1, 2, \ldots, K$ **do**
16:         compute $\mathcal{L}\big(q^{\mathrm{split}(k)}\big)$ via restricted iteration
17:         **if** $\mathcal{L}\big(q^{\mathrm{split}(k)}\big) > \mathcal{L}(q)$ **then**
18:             $q \leftarrow q^{\mathrm{split}(k)}$
19:         **end if**
20:     **end for**
21: **end for**
---

uncertainty associated with the online method can be mitigated to some extent by using large mini-batches. Confidence intervals for the expected change in the variational objective can be computed, and might be useful in a more sophisticated acceptance rule. Note that our usage of a nested family of variational bounds is key to the accuracy and stability of our split-merge acceptance rules.

## 4 Experimental Results

To demonstrate the effectiveness of our split-merge moves, we compare three algorithms: batch variational inference (bHDP), online variational inference without split-merge (oHDP), and online variational inference with split-merge (oHDP-SM). On the NIPS corpus we also compare these three methods to collapsed Gibbs sampling (CGS) and the CRF-style oHDP model (oHDP-CRF) proposed by [4].[3] We test the models on one synthetic and two real datasets:

**Bars** A 20-topic bars dataset of the type introduced in [18], where topics can be viewed as bars on a $10 \times 10$ grid. The vocabulary size is 100, with a training set of 2000 documents and a test set of 200 documents, 250 words per document.

**NIPS** 1,740 documents from the Neural Information Processing Systems conference proceedings, 1988-2000. The vocabulary size is 13,649, and there are 2.3 million tokens in total. We randomly divide the corpus into a 1,392-document training set and a 348-document test set.

**New York Times** The New York Times Annotated Corpus[4] consists of over 1.8 million articles appearing in the New York Times between 1987 and 2007. The vocabulary is pruned to 8,000 words. We hold out a randomly selected subset of 5,000 test documents, and use the remainder for training.

All values of $K$ given for oHDP-SM models are initial values — the actual truncation levels fluctuate during inference. While the truncation level $K$ is different from the actual number of topics assigned non-negligible mass, the split-merge model tends to merge away unused topics, so these numbers are usually fairly close. Hyperparameters are initialized to consistent values across all algorithms and datasets, and learned via Newton-Raphson updates (or in the case of CGS, resampled). We use a constant learning rate across all online algorithms. As suggested by [4], we set $\rho_t = (\tau + t)^{-\kappa}$ where $\tau = 1$, $\kappa = 0.5$. Empirically, we found that slower learning rates could result in greatly reduced performance, across all models and datasets.

To compare algorithm performance, we use per-word heldout likelihood, similarly to the metrics of [3, 19, 4]. We randomly split each test document in $\mathcal{D}^{\text{test}}$ into 80%-20% pieces, $w_{j1}$ and $w_{j2}$. Then, using $\bar{\phi}$ as the variational expectation of the topics from training, we learn $\bar{\pi}_j$ on $w_{j1}$ and approximate the probability of $w_{j2}$ as $\prod_{w \in w_{j2}} \sum_k \bar{\pi}_{jk} \bar{\phi}_{kw}$. The overall test metric is then

$$\mathcal{E} = \frac{\sum_{j \in \mathcal{D}^{\text{test}}} \sum_{w \in w_{j2}} \log \left( \sum_k \bar{\pi}_{jk} \bar{\phi}_{kw} \right)}{\sum_{j \in \mathcal{D}^{\text{test}}} |w_{j2}|}$$

## 4.1 Bars

For the bars data, we initialize eight oHDP-SM runs with $K = \{2, 5, 10, 20, 40, 50, 80, 100\}$, eight runs of oHDP with $K = 20$, and eight runs with $K = 50$. As seen in Figure 2(a), the oHDP algorithm converges to local optima, while the oHDP-SM runs all converge to the global optimum. More importantly, all split-merge methods converge to the correct number of topics, while oHDP uses either too few or too many topics. Note that the data-driven split-merge procedure allows splitting and merging of topics to mostly cease once the inference has converged (Figure 2(d)).

## 4.2 NIPS

We compare oHDP-SM, oHDP, bHDP, oHDP-CRF, and CGS in Figure 2. Shown are two runs of oHDP-SM with $K = \{100, 300\}$, two runs each of oHDP and bHDP with $K = \{300, 1000\}$, and one run each of oHDP-CRF and CGS with $K = 300$. All the runs displayed are the best runs from a larger sample of trials. Since oHDP and bHDP will use only a subset of topics under the truncation, setting $K$ much higher results in comparable numbers of topics as oHDP-SM. We set $|\mathcal{S}| = 200$ for the online algorithms, and run all methods for approximately 40 hours of CPU time.

The non split-merge methods reach poor local optima relatively quickly, while the split-merge algorithms continue to improve. Notably, both oHDP-CRF and CGS perform much worse than any of our methods. It appears that the CRF model performs very poorly for small datasets, and CGS reaches a mode quickly but does not mix between modes. Even though the split-merge algorithms improve in part by adding topics, they are using their topics much more effectively (Figure 2(h)). We speculate that for the NIPS corpus especially, the reason that models achieve better predictive likelihoods with more topics is due to the bursty properties of text data [20]. Figure 3 illustrates the topic refinement and specialization which occurs in successful split proposals.

## 4.3 New York Times

As batch variational methods and samplers are not feasible for such a large dataset, we compare two runs of oHDP with $K = \{300, 500\}$ to a run of oHDP-SM with $K = 200$ initial topics. We also use a larger minibatch size of $|\mathcal{S}| = 10,000$; split-merge acceptance decisions can sometimes be unstable with overly small minibatches. Figure 2(c) shows an inherent problem with oHDP for very large datasets — when truncated to $K = 500$, the algorithms uses all of its available topics and exhibits overfitting. For the oHDP-SM, however, predictive likelihood improves over a substantially longer period and overfitting is greatly reduced.

# 5 Discussion

We have developed a novel split-merge online variational algorithm for the hierarchical DP. This approach leads to more accurate models and better predictive performance, as well as a model that is able to adapt the number of topics more freely than conventional approximations based on fixed truncations. Our moves are similar in spirit to split-merge samplers, but by evaluating their quality stochastically using streaming data, we can rapidly adapt model structure to large-scale datasets.

While many papers have tried to improve conventional mean field methods via higher-order variational expansions [21], local optima can make the resulting algorithms compare unfavorably to Monte Carlo methods [3]. Here we pursue the complementary goal of more robust, scalable optimization of simple variational objectives. Generalization of our approach to more complex hierarchies of DPs, or basic DP mixtures, is feasible. We believe similar online learning methods will prove effective for the combinatorial structures of other Bayesian nonparametric models.

**Acknowledgments** We thank Dae Il Kim for his assistance with the experimental results.

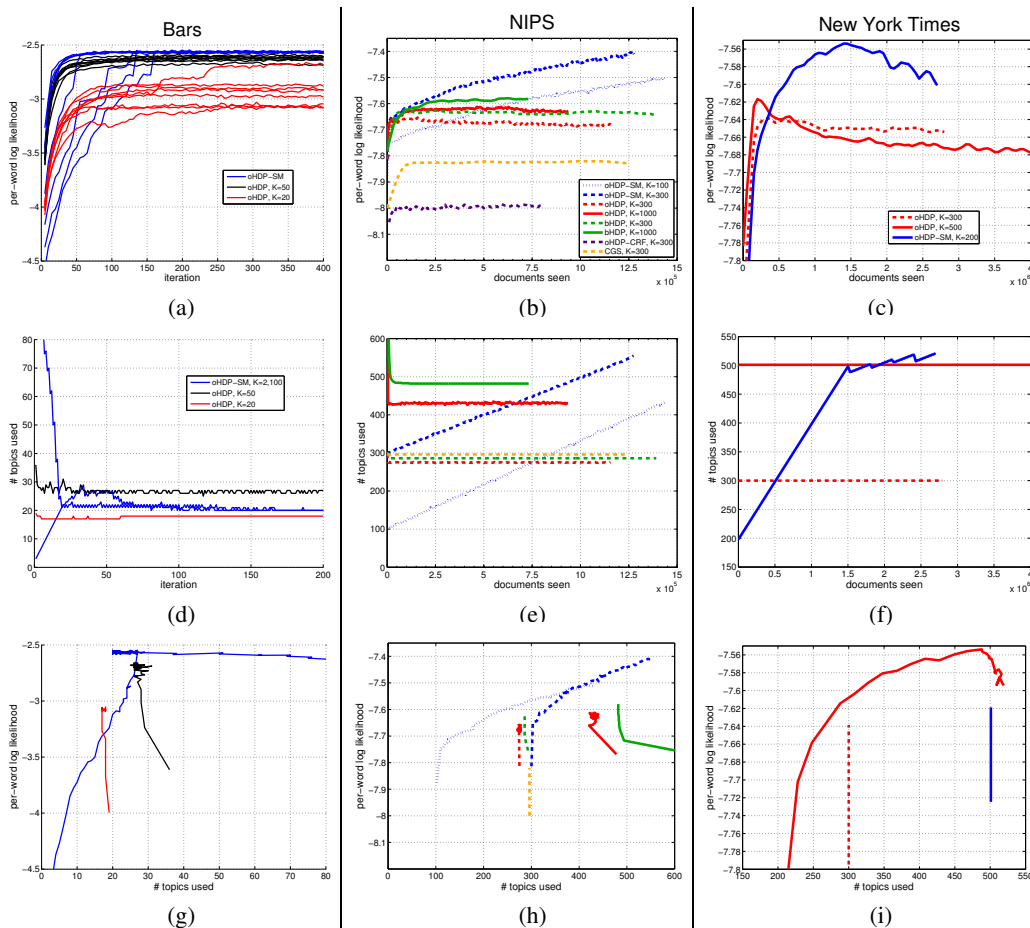

Figure 2: Trace plots of heldout likelihood and number of topics used. Across all datasets, common color indicates common algorithm, while for NIPS and New York Times, line type indicates different initializations. *Top:* Test log likelihood for each dataset. *Middle:* Number of topics used per iteration. *Bottom:* A plot of per-word log likelihood against number of topics used. Note particularly plot (h), where for every cardinality of used topics shown, there is a split-merge method outperforming a conventional method.

| Original topic | 40,000 | 80,000 | 120,000 | 160,000 | 200,000 | 240,000 |
|---|---|---|---|---|---|---|
| | patterns | patterns | patterns | patterns | patterns | patterns |
| | pattern | pattern | pattern | pattern | pattern | pattern |
| | cortex | cortex | cortex | cortex | cortex | cortex |
| | neurons | neurons | neurons | neurons | neurons | responses |
| | neuronal | neuronal | responses | responses | responses | types |
| patterns | responses | responses | neuronal | type | type | type |
| pattern | single | single | single | behavioral | behavioral | behavioral |
| cortex | inputs | temporal | type | types | types | form |
| neurons | temporal | inputs | number | neuronal | form | neurons |
| neuronal | activation | type | temporal | single | neuronal | areas |
| single | patterns | neuronal | neuronal | neuronal | neuronal | neuronal |
| responses | neuronal | patterns | neurons | dendritic | dendritic | dendritic |
| inputs | pattern | pattern | activation | peak | fire | postsynaptic |
| type | neurons | neurons | cortex | activation | peak | fire |
| activation | cortex | cortex | dendrite | cortex | activation | cortex |
| | inputs | activation | preferred | pyramidal | msec | activation |
| | activation | dendrite | patterns | msec | pyramidal | peak |
| | type | inputs | peak | fire | cortex | msec |
| | preferred | peak | pyramidal | dendrites | postsynaptic | pyramidal |
| | peak | preferred | inputs | inputs | inputs | inputs |

Figure 3: The evolution of a split topic. The left column shows the topic directly prior to the split. After 240,000 more documents have been analyzed, subtle differences become apparent: the top topic covers terms relating to general neuronal behavior, while the bottom topic deals more specifically with neuron firing.

## Footnotes

[1] We expect $\beta$ to have small posterior variance in large datasets, and using a point estimate $\beta^*$ simplifies variational derivations for our direct assignment formulation. As empirically explored for the HDP-PCFG [15], updates to the global topic weights have much less predictive impact than improvements to topic distributions.

[2]Technically, we replace topic $k$ with topic $k'$ and add $k''$ as a new topic. In practice, we found that the order of topics in the global stick-breaking distribution had little effect on overall algorithm performance.

[3]For CGS we use the code available at http://www.gatsby.ucl.ac.uk/~ywteh/research/npbayes/npbayes-r21.tgz, and for oHDP-CRF we use the code at http://www.cs.princeton.edu/~chongw/software/onlinehdp.tar.gz.

[4]http://www.ldc.upenn.edu/Catalog/catalogEntry.jsp?catalogId=LDC2008T19

# References

[1] Y.W. Teh, M. Jordan, and M. Beal. Hierarchical Dirichlet processes. *JASA*, 2006.

[2] D. Blei and M. Jordan. Variational methods for Dirichlet process mixtures. *Bayesian Analysis*, 1:121–144, 2005.

[3] Y.W. Teh, K. Kurihara, and M. Welling. Collapsed variational inference for HDP. *NIPS*, 2008.

[4] C. Wang, J. Paisley, and D. Blei. Online variational inference for the hierarchical Dirichlet process. *AISTATS*, 2011.

[5] M. Hoffman, D. Blei, and F. Bach. Online learning for latent Dirichlet allocation. *NIPS*, 2010.

[6] D. Blei, A. Ng, and M. Jordan. Latent Dirichlet allocation. *JMLR*, 2003.

[7] S. Jain and R. Neal. A split-merge Markov chain Monte Carlo procedure for the Dirichlet process mixture model. *Journal of Computational and Graphical Statistics*, 13:158–182, 2004.

[8] D.B. Dahl. Sequentially-allocated merge-split sampler for conjugate and nonconjugate Dirichlet process mixture models. Technical report, Texas A&M University, 2005.

[9] C. Wang and D. Blei. A split-merge MCMC algorithm for the hierarchical Dirichlet process. *ArXiv e-prints*, January 2012.

[10] N. Ueda, R. Nakano, Z. Ghahramani, and G. Hinton. SMEM algorithm for mixture models. *Neural Computation*, 2000.

[11] K. Kurihara and M. Welling. Bayesian K-means as a 'Maximization-Expectation' algorithm. *SIAM conference on data mining SDM06*, 2006.

[12] N. Ueda and Z. Ghahramani. Bayesian model search for mixture models based on optimizing variational bounds. *Neural Networks*, 15, 2002.

[13] Z. Ghahramani and M. Beal. Variational inference for Bayesian mixtures of factor analysers. *NIPS*, 2000.

[14] M. Jordan, Z. Ghahramani, T. Jaakkola, and L. Saul. Introduction to variational methods for graphical models. *Machine Learning*, 1999.

[15] P. Liang, S. Petrov, D. Klein, and M. Jordan. The infinite PCFG using hierarchical Dirichlet processes. *Empirical Methods in Natural Language Processing*, 2007.

[16] K. Kurihara, M. Welling, and N. Vlassis. Accelerated variational Dirichlet process mixtures. *NIPS*, 2007.

[17] D. Blei and C. Wang. Variational inference for the nested Chinese restaurant process. *NIPS*, 2009.

[18] T. L. Griffiths and M. Steyvers. Finding scientific topics. *PNAS*, 101:5228–5235, 2004.

[19] A. Asuncion, M. Welling, P. Smyth, and Y.W. Teh. On smoothing and inference for topic models. *UAI*, 2009.

[20] G. Doyle and C. Elkan. Accounting for word burstiness in topic models. *ICML*, 2009.

[21] M. J. Wainwright and M. I. Jordan. Graphical models, exponential families, and variational inference. *Foundations and Trends in Machine Learning*, 1:1–305, 2008.

